# Online Markov Decision Processes under Bandit Feedback

**Gergely Neu**[*†]
[*]Department of Computer Science and
Information Theory, Budapest University of
Technology and Economics, Hungary
neu.gergely@gmail.com

**András György**
[†]Machine Learning Research Group
MTA SZTAKI Institute for Computer
Science and Control, Hungary
gya@szit.bme.hu

**Csaba Szepesvári**
Department of Computing Science,
University of Alberta, Canada
szepesva@ualberta.ca

**András Antos**
Machine Learning Research Group
MTA SZTAKI Institute for Computer
Science and Control, Hungary
antos@szit.bme.hu

## Abstract

We consider online learning in finite stochastic Markovian environments where in each time step a new reward function is chosen by an oblivious adversary. The goal of the learning agent is to compete with the best stationary policy in terms of the total reward received. In each time step the agent observes the current state and the reward associated with the last transition, however, the agent does not observe the rewards associated with other state-action pairs. The agent is assumed to know the transition probabilities. The state of the art result for this setting is a no-regret algorithm. In this paper we propose a new learning algorithm and, assuming that stationary policies mix uniformly fast, we show that after $T$ time steps, the expected regret of the new algorithm is $\mathcal{O}\left(T^{2/3}(\ln T)^{1/3}\right)$, giving the first rigorously proved regret bound for the problem.

## 1 Introduction

We consider online learning in finite Markov decision processes (MDPs) with a fixed, known dynamics. The formal problem definition is as follows: An agent navigates in a finite stochastic environment by selecting actions based on the states and rewards experienced previously. At each time instant the agent observes the reward associated with the last transition and the current state, that is, at time $t+1$ the agent observes $r_t(\mathbf{x}_t, \mathbf{a}_t)$, where $\mathbf{x}_t$ is the state visited at time $t$ and $\mathbf{a}_t$ is the action chosen. The agent does not observe the rewards associated with other transitions, that is, the agent faces a *bandit* situation. The goal of the agent is to maximize its total expected reward $\hat{R}_T$ in $T$ steps. As opposed to the standard MDP setting, the reward function at each time step may be different. The only assumption about this sequence of reward functions $r_t$ is that they are chosen ahead of time, independently of how the agent acts. However, no statistical assumptions are made about the choice of this sequence. As usual in such cases, a meaningful performance measure for the agent is how well it can compete with a certain class of reference policies, in our case the set of all stationary policies: If $R_T^*$ denotes the expected total reward in $T$ steps that can be collected by choosing the best stationary policy (this policy can be chosen based on the full knowledge of the sequence $r_t$), the goal of learning can be expressed as minimizing the total expected regret, $\hat{L}_T = R_T^* - \hat{R}_T$.

In this paper we propose a new algorithm for this setting. Assuming that the stationary distributions underlying stationary policies exist, are unique and they are uniformly bounded away from zero and

that these policies mix uniformly fast, our main result shows that the total expected regret of our algorithm in $T$ time steps is $\mathcal{O}\left(T^{2/3}(\ln T)^{1/3}\right)$.

The first work that considered a similar online learning setting is due to Even-Dar et al. (2005, 2009). In fact, this is the work that provides the starting point for our algorithm and analysis. The major difference between our work and that of Even-Dar et al. (2005, 2009) is that they assume that the reward function is fully observed (i.e., in each time step the learning agent observes the whole reward function $r_t$), whereas we consider the bandit setting. The main result in these works is a bound on the total expected regret, which scales with the square root of the number of time steps under mixing assumptions identical to our assumptions. Another work that considered the full information problem is due to Yu et al. (2009) who proposed new algorithms and proved a bound on the expected regret of order $\mathcal{O}\left(T^{3/4+\varepsilon}\right)$ for arbitrary $\varepsilon \in (0, 1/3)$. The advantage of the algorithm of Yu et al. (2009) to that of Even-Dar et al. (2009) is that it is computationally less expensive, which, however, comes at the price of an increased bound on the regret. Yu et al. (2009) introduced another algorithm ("Q-FPL") and they have shown a sublinear ($o(T)$) almost sure bound on the regret.

All the works reviewed so far considered the full information case. The requirement that the full reward function must be given to the agent at every time step significantly limits their applicability. There are only three papers that we know of where the bandit situation was considered.

The first paper which falls into this category is due to Yu et al. (2009) who proposed an algorithm ("Exploratory FPL") for this setting and have shown an $o(T)$ almost sure bound on the regret.

Recently, Neu et al. (2010) gave $\mathcal{O}\left(\sqrt{T}\right)$ regret bounds for a special bandit setting when the agent interacts with a *loop-free episodic* environment. The algorithm and analysis in this work heavily exploits the specifics of these environments (i.e., that in the same episode no state can be visited twice) and so they do not generalize to our setting.

Another closely related work is due to Yu and Mannor (2009a,b) who considered the problem of online learning in MDPs where the transition probabilities may also change arbitrarily after each transition. This problem, however, is significantly different from ours and the algorithms studied are unsuitable for our setting. Further, the analysis in these papers seems to have gaps (see Neu et al., 2010). Thus, currently, the only result for the case considered in this paper is an asymptotic "no-regret" result.

The rest of the paper is organized as follows: The problem is laid out in Section 2, which is followed by a section about our assumptions (Section 3). The algorithm and the main result are given in Section 4, while a proof sketch of the latter is presented in Section 5.

## 2 Problem definition

Formally, a finite Markov Decision Process (MDP) $M$ is defined by a finite state space $\mathcal{X}$, a finite action set $\mathcal{A}$, a transition probability kernel $P : \mathcal{X} \times \mathcal{A} \times \mathcal{X} \to [0, 1]$, and a reward function $r : \mathcal{X} \times \mathcal{A} \to [0, 1]$. In time step $t \in \{1, 2, \ldots\}$, knowing the state $\mathbf{x}_t \in \mathcal{X}$, an agent acting in the MDP $M$ chooses an action $\mathbf{a}_t \in \mathcal{A}(\mathbf{x}_t)$ to be executed based on $(\mathbf{x}_t, r(\mathbf{a}_{t-1}, \mathbf{x}_{t-1}), \mathbf{a}_{t-1}, \mathbf{x}_{t-1}, \ldots, \mathbf{x}_2, r(\mathbf{a}_1, \mathbf{x}_1), \mathbf{a}_1, \mathbf{x}_1)$.[1] Here $\mathcal{A}(x) \subset \mathcal{A}$ is the set of admissible actions at state $x$. As a result of executing the chosen action the process moves to state $\mathbf{x}_{t+1} \in \mathcal{X}$ with probability $P(\mathbf{x}_{t+1}|\mathbf{x}_t, \mathbf{a}_t)$ and the agent receives reward $r(\mathbf{x}_t, \mathbf{a}_t)$. In the so-called average-reward problem, the goal of the agent is to maximize the average reward received over time. For a more detailed introduction the reader is referred to, for example, Puterman (1994).

### 2.1 Online learning in MDPs

In this paper we consider the online version of MDPs when the reward function is allowed to change arbitrarily. That is, instead of a single reward function $r$, a sequence of reward functions $\{r_t\}$ is given. This sequence is assumed to be fixed ahead of time, and, for simplicity, we assume that $r_t(x, a) \in [0, 1]$ for all $(x, a) \in \mathcal{X} \times \mathcal{A}$ and $t \in \{1, 2, \ldots\}$. No other assumptions are made about this sequence.

The learning agent is assumed to know the transition probabilities $P$, but is not given the sequence $\{r_t\}$. The protocol of interaction with the environment is unchanged: At time step $t$ the agent receives $\mathbf{x}_t$ and then selects an action $\mathbf{a}_t$ which is sent to the environment. In response, the reward $r_t(\mathbf{x}_t, \mathbf{a}_t)$ and the next state $\mathbf{x}_{t+1}$ are communicated to the agent. The initial state $\mathbf{x}_1$ is generated from a fixed distribution $P_0$.

The goal of the learning agent is to maximize its expected total reward

$$\hat{R}_T = \mathbb{E}\left[\sum_{t=1}^{T} r_t(\mathbf{x}_t, \mathbf{a}_t)\right].$$

An equivalent goal is to minimize the regret, that is, to minimize the difference between the expected total reward received by the best algorithm within some reference class and the expected total reward of the learning algorithm. In the case of MDPs a reasonable reference class, used by various previous works (e.g., Even-Dar et al., 2005, 2009; Yu et al., 2009) is the class of stationary stochastic policies.[2] A stationary stochastic policy, $\pi$, (or, in short: a policy) is a mapping $\pi : \mathcal{A} \times \mathcal{X} \to [0, 1]$, where $\pi(a|x) \equiv \pi(a, x)$ is the probability of taking action $a$ in state $x$. We say that a policy $\pi$ is followed in an MDP if the action at time $t$ is drawn from $\pi$, independently of previous states and actions given the current state $\mathbf{x}'_t$: $\mathbf{a}'_t \sim \pi(\cdot|\mathbf{x}'_t)$. The expected total reward while following a policy $\pi$ is defined as

$$R_T^{\pi} = \mathbb{E}\left[\sum_{t=1}^{T} r_t(\mathbf{x}'_t, \mathbf{a}'_t)\right].$$

Here $\{(\mathbf{x}'_t, \mathbf{a}'_t)\}$ denotes the trajectory that results from following policy $\pi$ from $\mathbf{x}'_1 \sim P_0$.

The *expected regret* (or *expected relative loss*) of the learning agent relative to the class of policies (in short, the regret) is defined as

$$\hat{L}_T = \sup_{\pi} R_T^{\pi} - \hat{R}_T,$$

where the supremum is taken over all (stochastic stationary) policies. Note that the optimal policy is chosen in hindsight, depending acausally on the reward function. If the regret of an agent grows sublinearly with $T$ then we can say that in the long run it acts as well as the best (stochastic stationary) policy (i.e., the average expected regret of the agent is asymptotically equal to that of the best policy).

## 3   Assumptions

In this section we list the assumptions that we make throughout the paper about the transition probability kernel (hence, these assumptions will not be mentioned in the subsequent results). In addition, recall that we assume that the rewards are bound to $[0, 1]$.

Before describing the assumptions, a few more definitions are needed: Let $\pi$ be a stationary policy. Define $P^{\pi}(x'|x) = \sum_a \pi(a|x) P(x'|x, a)$. We will also view $P^{\pi}$ as a matrix: $(P^{\pi})_{x,x'} = P^{\pi}(x'|x)$, where, without loss of generality, we assume that $\mathcal{X} = \{1, 2, \ldots, |\mathcal{X}|\}$. In general, distributions will also be treated as *row* vectors. Hence, for a distribution $\mu$, $\mu P^{\pi}$ is the distribution over $\mathcal{X}$ that results from using policy $\pi$ for one step from $\mu$ (i.e., the "next-state distribution" under $\pi$). Remember that the stationary distribution of a policy $\pi$ is a distribution $\mu$ which satisfies $\mu P^{\pi} = \mu$.

**Assumption A1**  *Every policy $\pi$ has a well-defined unique stationary distribution $\mu^{\pi}$.*

**Assumption A2**  *The stationary distributions are uniformly bounded away from zero:* $\inf_{\pi,x} \mu^{\pi}(x) \geq \beta$ *for some $\beta > 0$.*

**Assumption A3**  *There exists some fixed positive $\tau$ such that for any two arbitrary distributions $\mu$ and $\mu'$ over $\mathcal{X}$,*

$$\sup_{\pi} \|(\mu - \mu')P^{\pi}\|_1 \leq e^{-1/\tau}\|\mu - \mu'\|_1,$$

*where $\|\cdot\|_1$ is the 1-norm of vectors:* $\|v\|_1 = \sum_i |v_i|$.

Note that Assumption A3 implies Assumption A1. The quantity $\tau$ is called the *mixing time* underlying $P$ by Even-Dar et al. (2009) who also assume A3.

## 4  Learning in online MDPs under bandit feedback

In this section we shall first introduce some additional, standard MDP definitions, which we will be used later. That these are well-defined follows from our assumptions on $P$ and from standard results to be found, for example, in the book by Puterman (1994). After the definitions, we specify our algorithm. The section is finished by the statement of our main result concerning the performance of the proposed algorithm.

### 4.1  Preliminaries

Fix an arbitrary policy $\pi$ and $t \geq 1$. Let $\{(\mathbf{x}'_s, \mathbf{a}'_s)\}$ be the random trajectory generated by $\pi$ and the transition probability kernel $P$. Define, $\rho_t^\pi$, the *average reward per stage* corresponding to $\pi$, $P$ and $r_t$ by

$$\rho_t^\pi = \lim_{S \to \infty} \frac{1}{S} \sum_{s=0}^{S} \mathbb{E}[r_t(\mathbf{x}'_s, \mathbf{a}'_s)] \,.$$

An alternative expression for $\rho_t^\pi$ is $\rho_t^\pi = \sum_x \mu^\pi(x) \sum_a \pi(a|x) r_t(x, a)$, where $\mu^\pi$ is the stationary distribution underlying $\pi$. Let $q_t^\pi$ be the action-value function of $\pi$, $P$ and $r_t$ and $v_t^\pi$ be the corresponding state-value function. These can be uniquely defined as the solutions of the following Bellman equations:

$$q_t^\pi(x, a) = r_t(x, a) - \rho_t^\pi + \sum_{x'} P(x'|x, a) v_t^\pi(x'), \qquad v_t^\pi(x) = \sum_a \pi(a|x) q_t^\pi(x, a).$$

Now, consider the trajectory $\{(\mathbf{x}_t, \mathbf{a}_t)\}$ underlying a learning agent, where $\mathbf{x}_1$ is randomly chosen from $P_0$, and define

$$\mathbf{u}_t = (\,\mathbf{x}_1, \mathbf{a}_1, r_1(\mathbf{x}_1, \mathbf{a}_1),\ \mathbf{x}_2, \mathbf{a}_2, r_2(\mathbf{x}_2, \mathbf{a}_2),\ \ldots,\ \mathbf{x}_t, \mathbf{a}_t, r_t(\mathbf{x}_t, \mathbf{a}_t)\,)$$

and $\boldsymbol{\pi}_t(a|x) = \mathbb{P}[\mathbf{a}_t = a | \mathbf{u}_{t-1}, \mathbf{x}_t = x]$. That is, $\boldsymbol{\pi}_t$ denotes the policy followed by the agent at time step $t$ (which is computed based on past information and is therefore random). We will use the following notation:

$$\mathbf{q}_t = q_t^{\boldsymbol{\pi}_t}, \qquad\qquad \mathbf{v}_t = v_t^{\boldsymbol{\pi}_t}, \qquad\qquad \boldsymbol{\rho}_t = \rho_t^{\boldsymbol{\pi}_t}.$$

Note that $\mathbf{q}_t, \mathbf{v}_t$ satisfy the Bellman equations underlying $\boldsymbol{\pi}_t$, $P$ and $r_t$.

For reasons to be made clear later in the paper, we shall need the state distribution at time step $t$ given that we start from the state-action pair $(x, a)$ at time $t - N$, conditioned on the policies used between time steps $t - N$ and $t$:

$$\boldsymbol{\mu}_{t,x,a}^N(x') \overset{\text{def}}{=} \mathbb{P}\left[\mathbf{x}_t = x' \mid \mathbf{x}_{t-N} = x, \mathbf{a}_{t-N} = a, \boldsymbol{\pi}_{t-N+1}, \ldots, \boldsymbol{\pi}_{t-1}\right], \qquad x, x' \in \mathcal{X}, a \in \mathcal{A}.$$

It will be useful to view $\boldsymbol{\mu}_t^N$ as a matrix of dimensions $|\mathcal{X} \times \mathcal{A}| \times |\mathcal{X}|$. Thus, $\boldsymbol{\mu}_{t,x,a}^N(\cdot)$ will be viewed as one row of this matrix. To emphasize the conditional nature of this distribution, we will also use $\boldsymbol{\mu}_t^N(\cdot|x, a)$ instead of $\boldsymbol{\mu}_{t,x,a}^N(\cdot)$.

### 4.2  The algorithm

Our algorithm is similar to that of Even-Dar et al. (2009) in that we use an expert algorithm in each state. Since in our case the full reward function $r_t$ is not observed, the agent uses an estimate of it. The main difficulty is to come up with an unbiased estimate of $r_t$ with a controlled variance. Here we propose to use the following estimate:

$$\hat{\mathbf{r}}_t(x, a) = \begin{cases} \frac{r_t(x,a)}{\boldsymbol{\pi}_t(a|x)\boldsymbol{\mu}_t^N(x|\mathbf{x}_{t-N}, \mathbf{a}_{t-N})} & \text{if } (x, a) = (\mathbf{x}_t, \mathbf{a}_t) \\ 0 & \text{otherwise,} \end{cases} \tag{1}$$

where $t \geq N+1$. Define $\hat{\mathbf{q}}_t$, $\hat{\mathbf{v}}_t$ and $\hat{\rho}$ as the solution to the Bellman equations underlying the average reward MDP defined by $(P, \boldsymbol{\pi}_t, \hat{\mathbf{r}}_t)$:

$$
\hat{\mathbf{q}}_t(x,a) = \hat{\mathbf{r}}_t(x,a) - \hat{\rho}_t + \sum_{x'} P(x'|x,a)\hat{\mathbf{v}}_t(x'), \quad \hat{\mathbf{v}}_t(x) = \sum_a \boldsymbol{\pi}_t(a|x)\hat{\mathbf{q}}_t(x,a),
$$

$$
\hat{\rho}_t = \sum_{x,a} \mu^{\boldsymbol{\pi}_t}(x)\boldsymbol{\pi}_t(a|x)\hat{\mathbf{r}}_t(x,a). \tag{2}
$$

Note that if $N$ is sufficiently large and $\boldsymbol{\pi}_t$ changes sufficiently slowly then

$$
\boldsymbol{\mu}_t^N(x|\mathbf{x}_{t-N}, \mathbf{a}_{t-N}) > 0, \tag{3}
$$

almost surely, for arbitrary $x \in \mathcal{X}, t \geq N+1$. This fact will be shown in Lemma 4. Now, assume that $\boldsymbol{\pi}_t$ is computed based on $\mathbf{u}_{t-N}$, that is, $\boldsymbol{\pi}_t$ is measurable with respect to the $\sigma$-field $\sigma(\mathbf{u}_{t-N})$ generated by the history $\mathbf{u}_{t-N}$:

$$
\boldsymbol{\pi}_t \in \sigma(\mathbf{u}_{t-N}). \tag{4}
$$

Then also $\boldsymbol{\pi}_{t-1}, \ldots, \boldsymbol{\pi}_{t-N} \in \sigma(\mathbf{u}_{t-N})$ and $\boldsymbol{\mu}_t^N$ can be computed using

$$
\boldsymbol{\mu}_{t,x,a}^N = e_x P^a P^{\boldsymbol{\pi}_{t-N+1}} \cdots P^{\boldsymbol{\pi}_{t-1}}, \tag{5}
$$

where $P^a$ is the transition probability matrix when in every state action $a$ is used and $e_x$ is the unit row vector corresponding to $x$ (and we assumed that $\mathcal{X} = \{1, \ldots, |\mathcal{X}|\}$). Moreover, a simple but tedious calculation shows that (3) and (4) ensure the conditional unbiasedness of our estimates, that is,

$$
\mathbb{E}\left[\hat{\mathbf{r}}_t(x,a) | \mathbf{u}_{t-N}\right] = r_t(x,a). \tag{6}
$$

It then follows that

$$
\mathbb{E}[\hat{\rho}_t | \mathbf{u}_{t-N}] = \rho_t,
$$

and, hence, by the uniqueness of the solutions of the Bellman equations, we have, for all $(x,a) \in \mathcal{X} \times \mathcal{A}$,

$$
\mathbb{E}[\hat{\mathbf{q}}_t(x,a) | \mathbf{u}_{t-N}] = \mathbf{q}_t(x,a) \quad \text{and} \quad \mathbb{E}[\hat{\mathbf{v}}_t(x) | \mathbf{u}_{t-N}] = \mathbf{v}_t(x). \tag{7}
$$

As a consequence, we also have, for all $(x,a) \in \mathcal{X} \times \mathcal{A}, t \geq N+1$,

$$
\mathbb{E}[\hat{\rho}_t] = \mathbb{E}[\rho_t], \quad \mathbb{E}[\hat{\mathbf{q}}_t(x,a)] = \mathbb{E}[\mathbf{q}_t(x,a)], \quad \text{and} \quad \mathbb{E}[\hat{\mathbf{v}}_t(x)] = \mathbb{E}[\mathbf{v}_t(x)]. \tag{8}
$$

The bandit algorithm that we propose is shown as Algorithm 1. It follows the approach of Even-Dar et al. (2009) in that a bandit algorithm is used in each state which together determine the policy to be used. These bandit algorithms are fed with estimates of action-values for the current policy and the current reward. In our case these action-value estimates are $\hat{\mathbf{q}}_t$ defined earlier, which are based on the reward estimates $\hat{\mathbf{r}}_t$. A major difference is that the policy computed based on the most recent action-value estimates is used only $N$ steps later. This delay allows us to construct unbiased estimates of the rewards. Its price is that we need to store $N$ policies (or weights, leading to the policies), thus, the memory needed by our algorithm scales with $N|\mathcal{A}||\mathcal{X}|$. The computational complexity of the algorithm is dominated by the cost of computing $\hat{\mathbf{r}}_t$ (and, in particular, by the cost of computing $\boldsymbol{\mu}_t^N(\cdot|\mathbf{x}_{t-N}, \mathbf{a}_{t-N})$). The cost of this is $\mathcal{O}\left(N|\mathcal{A}||\mathcal{X}|^3\right)$. In addition to the need of dealing with the delay, we also need to deal with the fact that in our case $\mathbf{q}_t$ and $\hat{\mathbf{q}}_t$ can be both negative, which must be taken into account in the proper tuning of the algorithm's parameters.

## 4.3 Main result

Our main result is the following bound concerning the performance of Algorithm 1.

**Theorem 1.** *Let $N = \lceil \tau \ln T \rceil$,*

$$
\eta = T^{-2/3} \cdot (\ln|\mathcal{A}|)^{2/3} \cdot \left(\frac{4\tau + 8}{\beta}\left((2\tau+4)\tau|\mathcal{A}|\ln T + (3\tau+1)^2\right)\right)^{-1/3},
$$

$$
\gamma = T^{-1/3} \cdot (2\tau+4)^{-2/3} \cdot \left(\frac{2\ln|\mathcal{A}|}{\beta}\left((2\tau+4)\tau|\mathcal{A}|\ln T + (3\tau+1)^2\right)\right)^{1/3}.
$$

---

**Algorithm 1** Algorithm for the online bandit MDP.

---

Set $N \geq 1$, $\mathbf{w}_1(x,a) = \mathbf{w}_2(x,a) = \cdots = \mathbf{w}_{2N}(x,a) = 1$, $\gamma \in (0,1)$, $\eta \in (0, \gamma]$.

For $t = 1, 2, \ldots, T$, repeat

    1. Set

$$\boldsymbol{\pi}_t(a|x) = (1 - \gamma)\frac{\mathbf{w}_t(x,a)}{\sum_b \mathbf{w}_t(x,b)} + \frac{\gamma}{|\mathcal{A}|}$$

    for all $(x,a) \in \mathcal{X} \times \mathcal{A}$.

    2. Draw an action $\mathbf{a}_t$ randomly, according to the policy $\boldsymbol{\pi}_t(\cdot|\mathbf{x}_t)$.

    3. Receive reward $r_t(\mathbf{x}_t, \mathbf{a}_t)$ and observe $\mathbf{x}_{t+1}$.

    4. If $t \geq N + 1$

        (a) Compute $\boldsymbol{\mu}_t^N(x|\mathbf{x}_{t-N}, \mathbf{a}_{t-N})$ for all $x \in \mathcal{X}$ using (5).

        (b) Construct estimates $\hat{\mathbf{r}}_t$ using (1) and compute $\hat{\mathbf{q}}_t$ using (2).

        (c) Set $\mathbf{w}_{t+N}(x,a) = \mathbf{w}_{t+N-1}(x,a)e^{\eta\hat{\mathbf{q}}_t(x,a)}$ for all $(x,a) \in \mathcal{X} \times \mathcal{A}$.

---

*Then the regret can be bounded as*

$$\hat{L}_T \leq 3\,T^{2/3} \cdot \left(\frac{(4\tau + 8)\ln|\mathcal{A}|}{\beta}\left((2\tau + 4)\tau|\mathcal{A}|\ln T + (3\tau + 1)^2\right)\right)^{1/3} + \mathcal{O}\left(T^{1/3}\right).$$

It is interesting to note that, similarly to the regret bound of Even-Dar et al. (2009), the main term of the regret bound does not directly depend on the size of the state space, but it depends on it only through $\beta$ and the mixing time $\tau$, defined in Assumptions A2 and A3, respectively; however, we also need to note that $\beta > 1/|\mathcal{X}|$. While the theorem provides the first rigorously proved finite sample regret bound for the online bandit MDP problem, we suspect that the given convergence rate is not sharp in the sense that it may be possible, in agreement with the standard bandit lower bound of Auer et al. (2002), to give an algorithm with an $\mathcal{O}\left(\sqrt{T}\right)$ regret (up to some logarithmic factors).

The proof of the theorem is similar to the proof of a similar bound done for the full-information case by Even-Dar et al. (2009). Clearly, it suffices to bound $R_T^\pi - \hat{R}_T$ for an arbitrary fix policy $\pi$. We use the following decomposition of this difference (also used by Even-Dar et al., 2009):

$$R_T^\pi - \hat{R}_T = \left(R_T^\pi - \sum_{t=1}^T \rho_t^\pi\right) + \left(\sum_{t=1}^T \rho_t^\pi - \sum_{t=1}^T \boldsymbol{\rho}_t\right) + \left(\sum_{t=1}^T \boldsymbol{\rho}_t - \hat{R}_T\right). \tag{9}$$

The first term is bounded using the following standard MDP result.

**Lemma 1** (Even-Dar et al., 2009). *For any policy $\pi$ and any $T \geq 1$ it holds that* $\left(R_T^\pi - \sum_{t=1}^T \rho_t^\pi\right) \leq 2(\tau + 1)$.

Hence, it remains to bound the expectation of the other terms, which is done in the following two propositions.

**Proposition 1.** *Let $N \geq \lceil\tau\ln T\rceil$. For any policy $\pi$ and for all $T$ large enough, we have*

$$\sum_{t=1}^T \mathbb{E}\left[\rho_t^\pi - \boldsymbol{\rho}_t\right]$$

$$\leq (4\tau + 10)N + \frac{\ln|\mathcal{A}|}{\eta} + (2\tau + 4)\,T\left(\gamma + \frac{2\eta}{\beta}|\mathcal{A}|\Big(N\left(1/\gamma + 4\tau + 6\right) + (e - 2)(2\tau + 4)\Big)\right).$$

**Proposition 2.** *Let $N \geq \lceil\tau\ln T\rceil$. For any $T$ large enough,*

$$\sum_{t=1}^T \mathbb{E}\left[\boldsymbol{\rho}_t\right] - \hat{R}_T \leq T\frac{2\eta}{\beta}\left(\frac{1}{\gamma} + 4\tau + 6\right)(3\tau + 1)^2 + 2Te^{-N/\tau} + 2N. \tag{10}$$

Note that the choice of $N$ ensures that the second term in (10) becomes $\mathcal{O}(1)$.

The proofs are broken into a number of statements presented in the next section. Due to space constraints we present proof sketches only; the full proofs are presented in the extended version of the paper.

# 5 Analysis

## 5.1 General tools

First, we show that if the policies that we follow up to time step $t$ change slowly, $\boldsymbol{\mu}_t^N$ is "close" to $\mu^{\boldsymbol{\pi}_t}$:

**Lemma 2.** *Let* $1 \leq N < t \leq T$ *and* $c > 0$ *be such that* $\max_x \sum_a |\boldsymbol{\pi}_{s+1}(a|x) - \boldsymbol{\pi}_s(a|x)| \leq c$ *holds for* $1 \leq s \leq t - 1$. *Then we have*

$$\max_{x,a} \sum_{x'} \left| \boldsymbol{\mu}_{t,x,a}^N(x') - \mu^{\boldsymbol{\pi}_t}(x') \right| \leq c \left(3\tau + 1\right)^2 + 2e^{-N/\tau}.$$

In the next two lemmas we compute the rate of change of the policies produced by **Exp3** and show that for a large enough value of $N$, $\boldsymbol{\mu}_{t,x,a}^N$ can be uniformly bounded form below by $\beta/2$.

**Lemma 3.** *Assume that for some* $N + 1 \leq t \leq T$, $\boldsymbol{\mu}_{t,\mathbf{x}_{t-N},\mathbf{a}_{t-N}}^N(x') \geq \beta/2$ *holds for all states* $x'$. *Let* $c = \frac{2\eta}{\beta} \left( \frac{1}{\gamma} + 4\tau + 6 \right)$. *Then,*

$$\max_x \sum_a |\boldsymbol{\pi}_{t+N-1}(a|x) - \boldsymbol{\pi}_{t+N}(a|x)| \leq c. \tag{11}$$

The previous results yield the following result that show that by choosing the parameters appropriately, the policies will change slowly and $\boldsymbol{\mu}_t^N$ will be uniformly bounded away from zero.

**Lemma 4.** *Let* $c$ *be as in Lemma 3. Assume that* $c(3\tau + 1)^2 < \beta/2$, *and let*

$$N \geq \left\lceil \tau \ln \left( \frac{4}{\beta - 2c(3\tau + 1)^2} \right) \right\rceil. \tag{12}$$

*Then, for all* $N < t \leq T$, $x, x' \in \mathcal{X}$ *and* $a \in \mathcal{A}$, *we have* $\boldsymbol{\mu}_{t,x,a}^N(x') \geq \beta/2$ *and* $\max_{x'} \sum_{a'} |\boldsymbol{\pi}_{t+1}(a'|x') - \boldsymbol{\pi}_t(a'|x')| \leq c$.

This result is proved by first ensuring that $\boldsymbol{\mu}_t$ is uniformly lower bounded for $t = N + 1, \ldots, 2N$, which can be easily seen since the policies do not change in this period. For the rest of the time instants, one can proceed by induction, using Lemmas 2 and 3 in the inductive step.

## 5.2 Proof of Proposition 1

The statement is trivial for $T \leq N$. The following simple result is the first step in proving Proposition 1 for $T > N$.

**Lemma 5.** *(cf. Lemma 4.1 in Even-Dar et al., 2009) For any policy* $\pi$ *and* $t \geq 1$,

$$\rho_t^\pi - \boldsymbol{\rho}_t = \sum_{x,a} \mu^\pi(x) \pi(a|x) \left[ \mathbf{q}_t(x,a) - \mathbf{v}_t(x) \right].$$

For every $x, a$ define $\mathbf{Q}_T(x,a) = \sum_{t=N+1}^T \mathbf{q}_t(x,a)$ and $\mathbf{V}_T(x) = \sum_{t=N+1}^T \mathbf{v}_t(x)$. The preceding lemma shows that in order to prove Proposition 1, it suffices to prove an upper bound on $\mathbb{E} \left[ \mathbf{Q}_T(x,a) - \mathbf{V}_T(x) \right]$.

**Lemma 6.** *Let* $c$ *be as in Lemma 3. Assume that* $\gamma \in (0,1)$, $c(3\tau + 1)^2 < \beta/2$, $N \geq \left\lceil \tau \ln \left( \frac{4}{\beta - 2c(3\tau+1)^2} \right) \right\rceil$, $0 < \eta \leq \frac{\beta}{2(1/\gamma + 2\tau + 3)}$, *and* $T > N$ *hold. Then, for all* $(x,a) \in \mathcal{X} \times \mathcal{A}$,

$$\mathbb{E} \left[ \mathbf{Q}_T(x,a) - \mathbf{V}_T(x) \right]$$
$$\leq (4\tau + 8)N + \frac{\ln |\mathcal{A}|}{\eta} + (2\tau + 4) T \left( \gamma + \frac{2\eta}{\beta} |\mathcal{A}| \left( N \left(1/\gamma + 4\tau + 6\right) + (e - 2)(2\tau + 4) \right) \right).$$

*Proof sketch.* The proof essentially follows the original proof of Auer et al. (2002) concerning the regret bound of **Exp3**, although some details are more subtle in our case: our estimates have different properties than the ones considered in the original proof, and we also have to deal with the $N$-step delay.

Let

$$\hat{\mathbf{V}}_T^N(x) = \sum_{t=N+1}^{T-N+1} \sum_a \boldsymbol{\pi}_{t+N-1}(a|x)\hat{\mathbf{q}}_t(x,a) \quad \text{and} \quad \hat{\mathbf{Q}}_T^N(x,b) = \sum_{t=N+1}^{T-N+1} \hat{\mathbf{q}}_t(x,b).$$

Observe that although $\mathbf{q}_t(x,a)$ is not necessarily positive (in contrast to the rewards in the **Exp3** algorithm), one can prove that $\boldsymbol{\pi}_t(a|x)|\hat{\mathbf{q}}_t(x,a)| \leq \frac{4}{\beta}(\tau+2)$ and

$$\mathbb{E}\left[|\hat{\mathbf{q}}_t(x,a)|\right] \leq 2(\tau+2). \tag{13}$$

Similarly, it can be easily seen that the constraint on $\eta$ ensures that $\eta\hat{\mathbf{q}}_t(x,a) \leq 1$ for all $x,a,t$. Then, following the proof of Auer et al. (2002), we can show that

$$\hat{\mathbf{V}}_T^N(x) \geq (1-\gamma)\hat{\mathbf{Q}}_T^N(x,b) - \frac{\ln|\mathcal{A}|}{\eta} - \frac{4}{\beta}(\tau+2)\eta(e-2)\sum_{t=N+1}^{T-N+1}\sum_a |\hat{\mathbf{q}}_t(x,a)|. \tag{14}$$

Next, since the policies satisfy $\max_x \sum_a |\boldsymbol{\pi}_{s+1}(a|x) - \boldsymbol{\pi}_s(a|x)| \leq c$ by Lemma 4, we can prove, using (8) and (13), that

$$\mathbb{E}\left[\hat{\mathbf{V}}_T^N(x)\right] \leq \mathbb{E}\left[\mathbf{V}_T(x)\right] + 2(\tau+2)N(cT|\mathcal{A}|+1).$$

Now, taking the expectation of both sides of (14) and using the bound on $\mathbb{E}\left[\hat{\mathbf{V}}_T^N(x)\right]$ we get

$$\mathbb{E}\left[\mathbf{V}_T(x)\right] \geq (1-\gamma)\mathbb{E}\left[\mathbf{Q}_T^N(x,b)\right] - \frac{\ln|\mathcal{A}|}{\eta} - \frac{4}{\beta}(\tau+2)\eta(e-2)\sum_{t=N+1}^{T-N+1}\sum_a \mathbb{E}\left[|\hat{\mathbf{q}}_t(x,a)|\right]$$
$$- 2(\tau+2)N(cT|\mathcal{A}|+1),$$

where we used that $\mathbb{E}\left[\hat{\mathbf{Q}}_T^N(x,b)\right] = \mathbb{E}\left[\mathbf{Q}_T^N(x,b)\right]$ by (8). Since $\mathbf{q}_t(x,b) \leq 2(\tau+2)$,

$$\mathbb{E}\left[\mathbf{Q}_T^N(x,b)\right] \leq \mathbb{E}\left[\mathbf{Q}_T(x,b)\right] + 2(\tau+2)N.$$

Combining the above results and using (13) again, then substituting the definition of $c$ yields the desired result. $\square$

*Proof of Proposition 1.* Under the conditions of the proposition, combining Lemmas 5-6 yields

$$\sum_{t=1}^T \mathbb{E}\left[\rho_t^\pi - \boldsymbol{\rho}_t\right]$$
$$\leq 2N + \sum_{x,a}\mu^\pi(x)\pi(a|x)\mathbb{E}\left[\mathbf{Q}_T(x,a) - \mathbf{V}_T(x)\right]$$
$$\leq (4\tau+10)N + \frac{\ln|\mathcal{A}|}{\eta} + (2\tau+4)T\left(\gamma + \frac{2\eta}{\beta}|\mathcal{A}|\left(N(1/\gamma + 4\tau + 6) + (e-2)(2\tau+4)\right)\right),$$

proving Proposition 1. $\square$

# Acknowledgments

This work was supported in part by the Hungarian Scientific Research Fund and the Hungarian National Office for Research and Technology (OTKA-NKTH CNK 77782), the PASCAL2 Network of Excellence under EC grant no. 216886, NSERC, AITF, the Alberta Ingenuity Centre for Machine Learning, the DARPA GALE project (HR0011-08-C-0110) and iCore.

## Footnotes

[1] We follow the convention that boldface letters denote random variables.

[2]This is a reasonable reference class because for a fixed reward function one can always find a member of it which maximizes the average reward per time step, see Puterman (1994).

# References

Auer, P., Cesa-Bianchi, N., Freund, Y., and Schapire, R. E. (2002). The nonstochastic multiarmed bandit problem. *SIAM J. Comput.*, 32(1):48–77.

Even-Dar, E., Kakade, S. M., and Mansour, Y. (2005). Experts in a Markov decision process. In Saul, L. K., Weiss, Y., and Bottou, L., editors, *Advances in Neural Information Processing Systems 17*, pages 401–408.

Even-Dar, E., Kakade, S. M., and Mansour, Y. (2009). Online Markov decision processes. *Mathematics of Operations Research*, 34(3):726–736.

Neu, G., György, A., and Szepesvári, C. (2010). The online loop-free stochastic shortest-path problem. In *COLT-10*.

Puterman, M. L. (1994). *Markov Decision Processes: Discrete Stochastic Dynamic Programming*. Wiley-Interscience.

Yu, J. Y. and Mannor, S. (2009a). Arbitrarily modulated Markov decision processes. In *Joint 48th IEEE Conference on Decision and Control and 28th Chinese Control Conference*. IEEE Press.

Yu, J. Y. and Mannor, S. (2009b). Online learning in Markov decision processes with arbitrarily changing rewards and transitions. In *GameNets'09: Proceedings of the First ICST international conference on Game Theory for Networks*, pages 314–322, Piscataway, NJ, USA. IEEE Press.

Yu, J. Y., Mannor, S., and Shimkin, N. (2009). Markov decision processes with arbitrary reward processes. *Mathematics of Operations Research*, 34(3):737–757.

